# High-Rank Matrix Completion and Clustering under Self-Expressive Models

**E. Elhamifar**[*]
College of Computer and Information Science
Northeastern University
Boston, MA 02115
`eelhami@ccs.neu.edu`

## Abstract

We propose efficient algorithms for simultaneous clustering and completion of incomplete high-dimensional data that lie in a union of low-dimensional subspaces. We cast the problem as finding a completion of the data matrix so that each point can be reconstructed as a linear or affine combination of a few data points. Since the problem is NP-hard, we propose a lifting framework and reformulate the problem as a group-sparse recovery of each incomplete data point in a dictionary built using incomplete data, subject to rank-one constraints. To solve the problem efficiently, we propose a rank pursuit algorithm and a convex relaxation. The solution of our algorithms recover missing entries and provides a similarity matrix for clustering. Our algorithms can deal with both low-rank and high-rank matrices, does not suffer from initialization, does not need to know dimensions of subspaces and can work with a small number of data points. By extensive experiments on synthetic data and real problems of video motion segmentation and completion of motion capture data, we show that when the data matrix is low-rank, our algorithm performs on par with or better than low-rank matrix completion methods, while for high-rank data matrices, our method significantly outperforms existing algorithms.

## 1 Introduction

High-dimensional data, which are ubiquitous in computer vision, image processing, bioinformatics and social networks, often lie in low-dimensional subspaces corresponding to different categories they belong to [1, 2, 3, 4, 5, 6]. Clustering and finding low-dimensional representations of data are important unsupervised learning problems with numerous applications, including data compression and visualization, image/video/costumer segmentation, collaborative filtering and more.

A major challenge in real problems is dealing with missing entries in data, due to sensor failure, ad-hoc data collection, or partial knowledge of relationships in a dataset. For instance, in estimating object motions in videos, the tracking algorithm may loose the track of features in some video frames [7]; in the image inpainting problem, intensity values of some pixels are missing due to sensor failure [8]; or in recommender systems, each user provides ratings for a limited number of products [9].

**Prior Work.** Existing algorithms that deal with missing entries in high-dimensional data can be divided into two main categories. The first group of algorithms assume that data lie in a single low-dimensional subspace. Probabilistic PCA (PPCA) [10] and Factor Analysis (FA) [11] optimize a non-convex function using Expectation Maximization (EM), estimating low-dimensional model parameters and missing entries of data in an iterative framework. However, their performance depends

---

[*]E. Elhamifar is an Assistant Professor in the College of Computer and Information Science, Northeastern University.

on initialization and degrades as the dimension of the subspace or the percentage of missing entries increases. Low-rank matrix completion algorithms, such as [12, 13, 14, 15, 16, 17] recover missing entries by minimizing the convex surrogate of the rank, i.e., nuclear norm, of the complete data matrix. When the underlying subspace is incoherent with standard basis vectors and missing entries locations are spread uniformly at random, they are guaranteed to recover missing entries.

The second group of algorithms addresses the more general and challenging scenario where data lie in a union of low-dimensional subspaces. The goals in this case are to recover missing entries and cluster data according to subspaces. Since the union of low-dimensional subspaces is often high/full-rank, methods in the first category are not effective. Mixture of Probabilistic PCA (MPPCA) [18, 19], Mixture of Factor Analyzers (MFA) [20] and K-GROUSE [21] address clustering and completion of multi-subspace data, yet suffer from dependence on initialization and perform poorly as the dimension/number of subspaces or the percentage of missing entires increases. On the other hand, [22] requires a polynomial number of data points in the ambient space dimension, which often cannot be met in high-dimensional datasets. Building on the unpublished abstract in [23], a clustering algorithm using expectation completion on the data kernel matrix was proposed in [24]. However, the algorithm only addresses clustering and the resulting non-convex optimization is dealt with using the heuristic approach of shifting eigenvalues of the Hessian to nonnegative values. [25] assumes that the observed matrix corresponds to applying a Lipschitz, monotonic function to a low-rank matrix. While an important generalization to low-rank regime, [25] cannot cover the case of multiple subspaces.

**Paper Contributions.** In this paper, we propose an efficient algorithm for the problem of simultaneous completion and clustering of incomplete data lying in a union of low-dimensional subspaces. Building on the Sparse Subspace Clustering (SSC) algorithm [26], we cast the problem as finding a completion of the data so that each complete point can be efficiently reconstructed using a few complete points from the same subspace. Since the formulation is non-convex and, in general, NP-hard, we propose a lifting scheme, where we cast the problem as finding a group-sparse representation of each incomplete data point in a modified dictionary, subject to a set of rank-one constraints. In our formulation, coefficients in groups correspond to pairwise similarities and missing entries of data. More specifically, our group-sparse recovery formulation finds a few incomplete data points that well reconstruct a given point and, at the same time, completes the selected data points in a globally consistent fashion. Our framework has several advantages over the state of the art:

– Unlike algorithms such as [22] that require a polynomial number of points in the ambient-space dimension, our framework needs about as many points as the subspace dimension not the ambient space. In addition, we do not need to know dimensions of subspaces a priori.

– While two-stage methods such as [24], which first obtain a similarity graph for clustering and then apply low-rank matrix completion to each cluster, fail when subspaces intersect or clustering fails, our method simultaneously recovers missing entries and builds a similarity matrix for clustering, hence, each goal benefits from the other. Moreover, in scenarios where a hard clustering does not exist, we can still recover missing entries.

– While we motivate and present our algorithm in the context of clustering and completion of multi-subspace data, our framework can address any task that relies on the self-expressiveness property of the data, e.g., column subset selection in the presence of missing data.

– By experiments on synthetic and real data, we show that our algorithm performs on par with or better than low-rank matrix completion methods when the data matrix is low-rank, while it significantly outperforms state-of-the-art clustering and completion algorithms when the data matrix is high-rank.

## 2 Problem Statement

Assume we have $L$ subspaces $\{\mathcal{S}_\ell\}_{\ell=1}^L$ of dimensions $\{d_\ell\}_{\ell=1}^L$ in an $n$-dimensional ambient space, $\mathbb{R}^n$. Let $\{\boldsymbol{y}_j\}_{j=1}^N$ denote a set of $N$ data points lying in the union of subspaces, where we observe only some entries of each $\boldsymbol{y}_j \triangleq \begin{bmatrix} y_{1j} & y_{2j} & \dots & y_{nj} \end{bmatrix}^\top$. Assume that we do not know a priori the bases for subspaces nor do we know which data points belong to which subspace. Given the incomplete data points, our goal is to recover missing entries and cluster the data into their underlying subspaces.

To set the notation, let $\Omega_j \subseteq \{1, \ldots, n\}$ and $\Omega_j^c$ denote, respectively, indices of observed and missing entries of $\boldsymbol{y}_j$. Let $\boldsymbol{U}_{\Omega_j} \in \mathbb{R}^{n \times |\Omega_j|}$ be the submatrix of the standard basis whose columns are indexed by $\Omega_j$. We denote by $\boldsymbol{P}_{\Omega_j} \in \mathbb{R}^{n \times n}$ the projection matrix onto the subspace spanned by $\boldsymbol{U}_{\Omega_j}$, i.e., $\boldsymbol{P}_{\Omega_j} \triangleq \boldsymbol{U}_{\Omega_j} \boldsymbol{U}_{\Omega_j}^\top$. Hence, $\boldsymbol{x}_j \triangleq \boldsymbol{U}_{\Omega_j^c}^\top \boldsymbol{y}_j \in \mathbb{R}^{|\Omega_j^c|}$ corresponds to the vector of missing entries of $\boldsymbol{y}_j$. We denote by $\bar{\boldsymbol{y}}_j$ an $n$-dimensional vector whose $i$-th coordinate is $y_{ij}$ for $i \in \Omega_j$ and is zero for $i \in \Omega_j^c$, i.e., $\bar{\boldsymbol{y}}_j \triangleq \boldsymbol{P}_{\Omega_j} \boldsymbol{y}_j \in \mathbb{R}^n$. We can write each $\boldsymbol{y}_j$ as the summation of two orthogonal vectors with observed and unobserved entries, i.e.,

$$\boldsymbol{y}_j = \boldsymbol{P}_{\Omega_j} \boldsymbol{y}_j + \boldsymbol{P}_{\Omega_j^c} \boldsymbol{y}_j = \bar{\boldsymbol{y}}_j + \boldsymbol{U}_{\Omega_j^c} \boldsymbol{U}_{\Omega_j^c}^\top \boldsymbol{y}_j = \bar{\boldsymbol{y}}_j + \boldsymbol{U}_{\Omega_j^c} \boldsymbol{x}_j. \tag{1}$$

Finally, we denote by $\boldsymbol{Y} \in \mathbb{R}^{n \times N}$ and $\bar{\boldsymbol{Y}} \in \mathbb{R}^{n \times N}$ matrices whose columns are complete data points $\{\boldsymbol{y}_j\}_{j=1}^N$ and zero-filled data $\{\bar{\boldsymbol{y}}_j\}_{j=1}^N$, respectively.

To address completion and clustering of multi-subspace data, we propose a unified framework to simultaneously recover missing entries and learn a similarity graph for clustering. To do so, we build on the SSC algorithm [26, 4], which we review next.

## 3 Sparse Subspace Clustering Review

The sparse subspace clustering (SSC) algorithm [26, 4] addresses the problem of clustering complete multi-subspace data. It relies on the observation that in a high-dimensional ambient space, while there are many ways that each data point $\boldsymbol{y}_j$ can be reconstructed using the entire dataset, a sparse representation selects a few data points from the underlying subspace of $\boldsymbol{y}_j$, since each point in $\mathcal{S}_\ell$ can be represented using $d_\ell$ data points, in general directions, from $\mathcal{S}_\ell$. This motivates solving[2]

$$\min_{\{c_{1j}, \ldots, c_{Nj}\}} \sum_{i=1}^N |c_{ij}| \quad \text{s.t.} \quad \sum_{i=1}^N c_{ij} \boldsymbol{y}_i = \boldsymbol{0}, \ c_{jj} = -1, \tag{2}$$

where the constraints express that each $\boldsymbol{y}_j$ should be written as a combination of other points. To infer clustering, one builds a similarity graph using sparse coefficients, by connecting nodes $i$ and $j$ of the graph, representing, respectively, $\boldsymbol{y}_i$ and $\boldsymbol{y}_j$, with an edge with the weight $w_{ij} = |c_{ij}| + |c_{ji}|$. Clustering of data is obtained then by applying spectral clustering [27] to the similarity graph.

While [4, 26, 28] show that, under appropriate conditions on subspace angles and data distribution, (2) is guaranteed to recover desired representations, the algorithm requires complete data points.

### 3.1 Naive Extensions of SSC to Deal with Missing Entries

In the presence of missing entries, the $\ell_1$-minimization in (2) becomes non-convex, since coefficients and a subset of data entries are both unknown. A naive approach is to solve (2) using zero-filled data points, $\{\bar{\boldsymbol{y}}_i\}_{i=1}^N$, to perform clustering and then apply standard matrix completion on each cluster. However, the drawback of this approach is that not only it does not take advantage of the known locations of missing entries, but also zero-filled data will no longer lie in original subspaces, and deviate more from subspaces as the percentage of missing entries increases. Hence, a sparse representation does not necessarily find points from the same subspace and spectral clustering fails.

An alternative approach to deal with incomplete data is to use standard low-rank matrix completion algorithms to recover missing values and then apply SSC to cluster data into subspaces. While this approach works when the union of subspaces is low-rank, its effectiveness diminishes as the number of subspaces or their dimensions increases and the data matrix becomes high/full-rank.

## 4 Sparse Subspace Clustering and Completion via Lifting

In this section, we propose an algorithm to recover missing entries and build a similarity graph for clustering, given observations $\{y_{ij}; \ i \in \Omega_j\}_{j=1}^N$ for $N$ data points lying in a union of subspaces.

## 4.1 SSC–Lifting Formulation

To address the problem, we start from the SSC observation that, given complete data $\{\boldsymbol{y}_j\}_{j=1}^N$, the solution of

$$\min_{\{c_{ij}\}} \sum_{j=1}^N \sum_{i=1}^N \mathrm{I}(|c_{ij}|) \quad \text{s.t.} \quad \sum_{i=1}^N c_{ij}\boldsymbol{y}_i = \boldsymbol{0}, \;\; c_{jj} = -1, \;\; \forall j \tag{3}$$

ideally finds a representation of each $\boldsymbol{y}_j$ as a linear combination of a few data points that lie in the same subspace as of $\boldsymbol{y}_j$. $\mathrm{I}(\cdot)$ denotes the indicator function, which is zero when its argument is zero and is one otherwise. Notice that, using (1), we can write each $\boldsymbol{y}_i$ as

$$\boldsymbol{y}_i = \bar{\boldsymbol{y}}_i + \boldsymbol{U}_{\Omega_i^c}\boldsymbol{x}_i = \begin{bmatrix} \bar{\boldsymbol{y}}_i & \boldsymbol{U}_{\Omega_i^c} \end{bmatrix} \begin{bmatrix} 1 \\ \boldsymbol{x}_i \end{bmatrix}, \tag{4}$$

where $\bar{\boldsymbol{y}}_i$ is the $i$-th data point whose missing entries are filled with zeros and $\boldsymbol{x}_i$ is the vector containing missing entries of $\boldsymbol{y}_i$. Thus, substituting (4) in the optimization (3), we would like to solve

$$\min_{\{c_{ij}\},\{\boldsymbol{x}_i\}} \sum_{j=1}^N \sum_{i=1}^N \mathrm{I}(|c_{ij}|) \quad \text{s.t.} \quad \sum_{i=1}^N \begin{bmatrix} \bar{\boldsymbol{y}}_i & \boldsymbol{U}_{\Omega_i^c} \end{bmatrix} \begin{bmatrix} c_{ij} \\ c_{ij}\boldsymbol{x}_i \end{bmatrix} = \boldsymbol{0}, \;\; c_{jj} = -1, \;\; \forall j. \tag{5}$$

Notice that matrices $\begin{bmatrix} \bar{\boldsymbol{y}}_i & \boldsymbol{U}_{\Omega_i^c} \end{bmatrix} \in \mathbb{R}^{n \times |\Omega_i^c|+1}$ are given and known while vectors $\begin{bmatrix} c_{ij} & c_{ij}\boldsymbol{x}_i^\top \end{bmatrix}^\top \in \mathbb{R}^{|\Omega_i^c|+1}$ are unknown. In fact, the optimization (5) has two sources of non-convexity: the $\ell_0$-norm in the objective function and the product of unknown variables $\{c_{ij}\}$ and $\{\boldsymbol{x}_i\}$ in the constraint.

To pave the way for an efficient algorithm, first we use the fact that the number of nonzero coefficients $c_{ij}$ is the same as the number of nonzero blocks $\begin{bmatrix} c_{ij} & c_{ij}\boldsymbol{x}_i^\top \end{bmatrix}^\top$, since $c_{ij}$ is nonzero if and only if $\begin{bmatrix} c_{ij} & c_{ij}\boldsymbol{x}_j^\top \end{bmatrix}^\top$ is nonzero. Thus, we can write (5) as the equivalent group-sparse optimization

$$\min_{\{c_{ij}\},\{\boldsymbol{x}_i\}} \sum_{j=1}^N \sum_{i=1}^N \mathrm{I}\left( \left\| \begin{bmatrix} c_{ij} \\ c_{ij}\boldsymbol{x}_i \end{bmatrix} \right\|_p \right) \quad \text{s.t.} \quad \sum_{i=1}^N \begin{bmatrix} \bar{\boldsymbol{y}}_i & \boldsymbol{U}_{\Omega_i^c} \end{bmatrix} \begin{bmatrix} c_{ij} \\ c_{ij}\boldsymbol{x}_i \end{bmatrix} = \boldsymbol{0}, \;\; c_{jj} = -1, \;\; \forall j, \tag{6}$$

where $\| \cdot \|_p$ denotes the $\ell_p$-norm for $p > 0$. Next, to deal with the non-convexity of the product of $c_{ij}$ and $\boldsymbol{x}_i$, we use the fact that for each $i \in \{1, \ldots, N\}$, the matrix

$$\boldsymbol{A}_i \triangleq \begin{bmatrix} c_{i1} & \cdots & c_{iN} \\ c_{i1}\boldsymbol{x}_i & \cdots & c_{iN}\boldsymbol{x}_i \end{bmatrix} = \begin{bmatrix} 1 \\ \boldsymbol{x}_i \end{bmatrix} \begin{bmatrix} c_{i1} & \cdots & c_{iN} \end{bmatrix}, \tag{7}$$

is of rank one, since it can be written as the outer product of two vectors. This motivates to use a lifting scheme where we define new optimization variables

$$\boldsymbol{\alpha}_{ij} \triangleq c_{ij}\boldsymbol{x}_i \in \mathbb{R}^{|\Omega_i^c|}, \tag{8}$$

and consider the group-sparse optimization program

$$\min_{\substack{\{c_{ij}\},\{\boldsymbol{\alpha}_{ij}\} \\ c_{jj}=-1,\forall j}} \sum_{j=1}^N \sum_{i=1}^N \mathrm{I}\left( \left\| \begin{bmatrix} c_{ij} \\ \boldsymbol{\alpha}_{ij} \end{bmatrix} \right\|_p \right) \quad \text{s.t.} \quad \sum_{i=1}^N \begin{bmatrix} \bar{\boldsymbol{y}}_i & \boldsymbol{U}_{\Omega_i^c} \end{bmatrix} \begin{bmatrix} c_{ij} \\ \boldsymbol{\alpha}_{ij} \end{bmatrix} = \boldsymbol{0}, \mathrm{rk}\left( \begin{bmatrix} c_{i1} & \cdots & c_{iN} \\ \boldsymbol{\alpha}_{i1} & \cdots & \boldsymbol{\alpha}_{iN} \end{bmatrix} \right) = 1, \forall i, j, \tag{9}$$

where we have replaced $c_{ij}\boldsymbol{x}_i$ with $\boldsymbol{\alpha}_{ij}$ and have introduced rank-one constraints. In fact, we show that one can recover the solution of (5) using (9) and vice versa.

**Proposition 1** *Given a solution $\{c_{ij}\}$ and $\{\boldsymbol{\alpha}_{ij}\}$ of (9), by computing $\boldsymbol{x}_i$'s via the factorization in (7), $\{c_{ij}\}$ and $\{\boldsymbol{x}_i\}$ is a solution of (5). Also, given a solution $\{c_{ij}\}$ and $\{\boldsymbol{x}_i\}$ of (5), $\{c_{ij}\}$ and $\{\boldsymbol{\alpha}_{ij} \triangleq c_{ij}\boldsymbol{x}_i\}$ would be a solution of (9).*

Notice that, we have transferred the non-convexity of the product $c_{ij}\boldsymbol{x}_i$ in (5) into a set of non-convex rank-one constraints in (9). However, as we will see next, (9) admits an efficient convex relaxation.

## 4.2 Relaxations and Extensions

The optimization program in (9) is, in general, NP-hard, due to the mixed $\ell_0/\ell_p$-norm in the objective function. It is non-convex due to both mixed $\ell_0/\ell_p$-norm and rank-one constraints. To solve (9), we first take the convex surrogate of the objective function, which corresponds to an $\ell_1/\ell_p$-norm [29, 30], where we drop the indicator function and, for $p \in \{2, \infty\}$, solve

$$\min_{\substack{\{c_{ij}, \boldsymbol{\alpha}_{ij}\} \\ \{c_{jj} = -1\}}} \lambda \sum_{j=1}^{N} \sum_{i=1}^{N} \left\| \begin{bmatrix} c_{ij} \\ \boldsymbol{\alpha}_{ij} \end{bmatrix} \right\|_p + \sum_{j=1}^{N} \rho \left( \sum_{i=1}^{N} \begin{bmatrix} \bar{\boldsymbol{y}}_i & \boldsymbol{U}_{\Omega_i^c} \end{bmatrix} \begin{bmatrix} c_{ij} \\ \boldsymbol{\alpha}_{ij} \end{bmatrix} \right) \text{ s.t. } \mathrm{rk} \left( \begin{bmatrix} c_{i1} & \cdots & c_{iN} \\ \boldsymbol{\alpha}_{i1} & \cdots & \boldsymbol{\alpha}_{iN} \end{bmatrix} \right) = 1, \forall i. \tag{10}$$

The nonnegative parameter $\lambda$ is a regularization parameter and the function $\rho(\cdot) \in \{\rho_e(\cdot), \rho_a(\cdot)\}$ enforces whether the reconstruction of each point should be exact or approximate, where

$$\rho_e(\boldsymbol{u}) \triangleq \begin{cases} +\infty & \text{if } \boldsymbol{u} \neq \boldsymbol{0} \\ 0 & \text{if } \boldsymbol{u} = \boldsymbol{0} \end{cases}, \qquad \rho_a(\boldsymbol{u}) \triangleq \frac{1}{2} \|\boldsymbol{u}\|_2^2. \tag{11}$$

More specifically, when dealing with missing entries from noise-free data, which perfectly lie in multiple subspaces, we enforce exact reconstruction by selecting $\rho(\cdot) = \rho_e(\cdot)$. On the other hand, when dealing with real data where observed entries are corrupted by noise, exact reconstruction is infeasible or comes at the price of losing the sparsity of the solution, which is undesired. Thus, to deal with noisy incomplete data, we consider approximate reconstruction by selecting $\rho(\cdot) = \rho_a(\cdot)$.

Notice that the objective function of (10) is convex for $p \geq 1$, while the rank-one constraints are non-convex. We can obtain a local solution, by solving (10) with an Alternating Direction Method of Multipliers (ADMM) framework using projection onto the set of rank-one matrices.

To obtain a convex algorithm, we use a nuclear-norm[3] relaxation [12, 14, 15] for the rank-one constraints, where we replace $\mathrm{rank}(\boldsymbol{A}_i) = 1$ with $\|\boldsymbol{A}_i\|_* \leq \tau$, for $\tau > 0$. In addition, to reduce the number of constraints and the complexity of the problem, we choose to bring the nuclear norm constraints into the objective function using a Lagrange multiple $\gamma > 0$. Hence, we propose to solve

$$\min_{\substack{\{c_{ij}, \boldsymbol{\alpha}_{ij}\} \\ \{c_{jj} = -1\}}} \lambda \sum_{j=1}^{N} \sum_{i=1}^{N} \left\| \begin{bmatrix} c_{ij} \\ \boldsymbol{\alpha}_{ij} \end{bmatrix} \right\|_p + \gamma \sum_{i=1}^{N} \left\| \begin{bmatrix} c_{i1} & \cdots & c_{iN} \\ \boldsymbol{\alpha}_{i1} & \cdots & \boldsymbol{\alpha}_{iN} \end{bmatrix} \right\|_* + \sum_{j=1}^{N} \rho \left( \sum_{i=1}^{N} \begin{bmatrix} \bar{\boldsymbol{y}}_i & \boldsymbol{U}_{\Omega_i^c} \end{bmatrix} \begin{bmatrix} c_{ij} \\ \boldsymbol{\alpha}_{ij} \end{bmatrix} \right), \tag{12}$$

which is convex for $p \geq 1$ and can be solved efficiently using convex solvers. Finally, using the solution of (10), we recover missing entries by finding the best rank-one factorization of each block $\boldsymbol{A}_i$ as in (7), which results in[4]

$$\hat{\boldsymbol{x}}_i = \frac{\sum_{j=1}^{N} c_{ij} \boldsymbol{\alpha}_{ij}}{\sum_{j=1}^{N} c_{ij}^2}. \tag{13}$$

In addition, we use the coefficients $\{c_{ij}\}$ to build a similarity graph with weights $w_{ij} = |c_{ij}| + |c_{ji}|$ and obtain clustering of data using graph partitioning. It is important to note that we do not need to know dimensions of subspaces a priori, since (10) automatically selects the appropriate number of data points from each subspace. Also, it is worth metioning that we can use $\sum_{j=1}^{N} \sum_{i=1}^{N} |c_{ij}|$ instead of the group-sparsity term in (10) and (12).

**Remark 1** *Notice that when all entries of all data points are observed, i.e., $\Omega_i^c = \varnothing$, the rank-one constraints in (9) are trivially satisfied. Hence, (10) and (12) with $\gamma = 0$ reduce to the $\ell_1$-minimization of SSC. In other words, our framework is a generalization of SSC, which simultaneously finds similarities and missing entries for incomplete data.*

Table 1 shows the stable rank[5] [31] of blocks $\boldsymbol{A}_i$ of the solution for the synthetic dataset explained in the experiments in Section 5. As the results show, the penalized optimization successfully recovers close to rank-one solutions for practical values of $\gamma$ and $\lambda$.

Table 1: Average stable-rank of matrices $\boldsymbol{A}_i$ for high-rank data, $n = 100$, $L = 12$, $d = 10$, $N = 600$, with $\rho = 0.4$, explained in section 5. Notice that rank of $\boldsymbol{A}_i$ is close to one, and as $\gamma$ increases, it gets closer to one.

|  | $\gamma = 0.001$ | $\gamma = 0.01$ | $\gamma = 0.1$ |
|---|---|---|---|
| $\lambda = 0.01$ | $1.015 \pm 0.005$ | $1.009 \pm 0.005$ | $1.004 \pm 0.002$ |
| $\lambda = 0.1$ | $1.021 \pm 0.007$ | $1.011 \pm 0.006$ | $1.006 \pm 0.003$ |

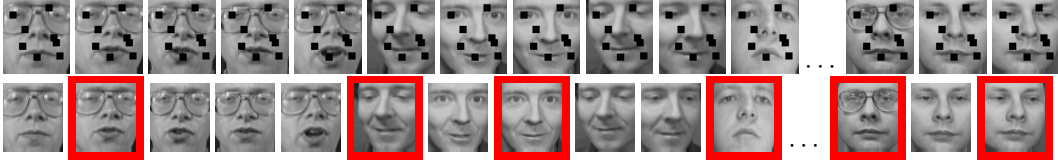

Figure 1: Subset selection and completion via lifting on the Olivetti face dataset. Top: faces from the dataset with missing entries. Bottom: solution of our method on the dataset. We successfully recover missing entries and, at the same time, select a subset of faces as representatives.

Notice that the mixed $\ell_1/\ell_p$-norm in the objective function of (10) and (12) promotes selecting a few nonzero coefficient blocks $\begin{bmatrix} c_{ij} & \boldsymbol{\alpha}_{ij}^\top \end{bmatrix}$. In other words, we find a representation of each incomplete data point using a few other incomplete data points, while, at the same time, find missing entries of the selected data points. On the other hand, rank constraints on the sub-blocks of the solution ensure that recovered missing entries are globally consistent, i.e., if a data point takes part in the reconstruction of multiple points, the associated missing entries in each representation are the same.

**Remark 2** *Our lifting framework can also deal with missing entries in other tasks that rely the on the self-expressiveness property, i.e., $\boldsymbol{y}_j = \sum_{i=1}^N c_{ij}\boldsymbol{y}_i$. Figure 1 shows results of the extension of our method to column subset selection [32, 33] with missing entries. In fact, simultaneously selecting a few data points that well reconstruct the entire dataset and recovering missing entires can be cast as a modification of (10) or (12), where we modify the first term in the objective function in order to select a few nonzero blocks, $\boldsymbol{A}_i$.*

## 5 Experiments

We study the performance of our algorithm for completion and clustering of synthetic and real data. We implement (10) and (12) with $\sum_{j=1}^N \sum_{i=1}^N |c_{ij}|$ instead of the group-sparsity term using the ADMM framework [34, 35]. Unless stated otherwise, we set $\lambda = 0.01$ and $\gamma = 0.1$. However, the results are stable for $\lambda \in [0.005, 0.05]$ and $\gamma \in [0.01, 0.5]$.

We compare our algorithm, SSC-Lifting, with MFA [20], K-Subspaces with Missing Entries (KSub-M) [21], Low-Rank Matrix Completion [13] followed by SSC (LRMC+SSC) or LSA [36] (LRMC+LSA), and SSC using Column-wise Expectation Completion (SSC-CEC) [24]. It is worth mentioning that in all experiments, we found that the performance of SSC-CEC is slightly better than SSC using zero-filled data. In addition, as reported in [21], KSub-M generally outperforms the high-rank matrix completion algorithm in [22], since the latter requires a very large number of samples, which becomes impractical in high-dimensional problems. We compute

$$\text{Clustering Error} = \frac{\text{\# Misclassified points}}{\text{\# All points}}, \quad \text{Completion Error} = \frac{\|\hat{\boldsymbol{Y}} - \boldsymbol{Y}\|_F}{\|\boldsymbol{Y}\|_F}, \qquad (14)$$

where $\boldsymbol{Y}$ and $\hat{\boldsymbol{Y}}$ denote, respectively, the true and recovered matrix and $\|\cdot\|_F$ is the Frobenius norm.

### 5.1 Synthetic Experiments

In this section, we evaluate the performance of different algorithms on synthetic data. We generate $L$ random $d$-dimensional subspaces in $\mathbb{R}^n$ and draw $N_g$ data points, at random, from each subspace. We consider two scenarios: 1) a low-rank data matrix whose columns lie in a union of low-dimensional subspaces; 2) a high rank data matrix whose columns lie in a union of low-dimensional subspaces. Unless stated otherwise, for low-rank matrices, we set $L = 3$ and $d = 5$, hence, $Ld = 15 < n = 100$, while for high-rank matrices, we set $L = 12$ and $d = 10$, hence, $Ld = 120 > n = 100$.

**Completion Performance.** We generate missing entries by selecting $\rho$ fraction of entries of the data matrix uniformly at random and dropping their values. The left and middle left plots in Figure 2

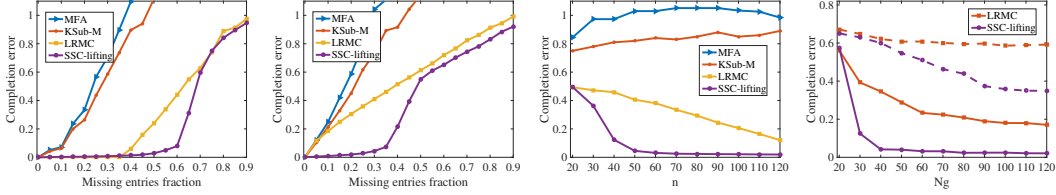

Figure 2: Completion errors of different algorithms as a function of $\rho$. Left: low-rank matrices. Middle left: high-rank matrices. Middle right: effect of the ambient space dimension, $n$. Right: effect of the number of data points in each subspace, $N_g$, for low-rank (solid lines) and high-rank (dashed lines) matrices.

show completion errors of different algorithms for low-rank and high-rank matrices, respectively, as a function of the fraction of missing entries, $\rho$. Notice that in both cases, MFA and KSub-M have high errors, which rapidly increase as $\rho$ increases, due to dependence on initialization and getting trapped in local optima. In both cases, SSC-lifting outperforms all methods across all values of $\rho$. Specifically, in the low-rank regime, while LRMC and SSC-lifting have almost zero error for $\rho \leq 0.35$, the performance of LRMC quickly degrades for larger $\rho$'s, while SSC-lifting performs well for $\rho \leq 0.6$. On the other hand, the performance of LRMC significantly degrades for the high-rank case, with a large gap to SSC-lifting, which performs well for $\rho < 0.45$. The middle right plot in Figure 2 demonstrates the effect of the ambient space dimension, $n$, for $L = 7$, $d = 5$, $N_g = 100$ and $\rho = 0.3$. Notice that errors of MFA and KSub-M increases as $n$ increases, due to larger number of local optima. LRMC has a large error for small values of $n$, where $n$ is smaller than or close to $Ld$, i.e., high-rank regime. As $n$ increases and matrices becomes low-rank, the error decreases. Notice that SSC-lifting for $n \geq 40$ has a low error, demonstrating its effectiveness in handling both low-rank and high-rank matrices. Finally, the right plot in Figure 2 demonstrates the effect of the number of points, $N_g$, for low and high rank matrices with $\rho = 0.5$. We do not show results of MFA and KSub-M, since they have large errors for all $N_g$. Notice that for all values of $N_g$, SSC-lifting obtains smaller errors than LRMC, verifying the effectiveness of sparsity principle to complete the data.

**Clustering Performance.** Next, we compare the clustering performance. To better study the effect of missing entries, we generate missing entries by selecting a fraction $\delta$ of data points and for each selected data point, we drop the values for a fraction $\rho$ of its entries, both uniformly at random. We change $\delta$ in $[0.1, 1.0]$ and $\rho$ in $[0.1, 0.9]$ and for each pair $(\rho, \delta)$, record the average clustering and completion errors over 20 trials, each with different random subspaces and data points. Figure 3 shows the clustering errors of different algorithms for low-rank (top row) and high-rank (bottom row) data matrices (completion errors provided in supplementary materials). In both cases, MFA performs poorly, due to local optima. While LRMC+SSC, SSC-CEC and SSC-Lifting perform similarly for low-rank matrices, SSC-Lifting performs best among all methods for high-rank matrices. In particular, when the percentage of missing entries, $\rho$, is more than 70%, SSC-Lifting performs significantly better than other algorithms. It is important to notice that for small values of $(\rho, \delta)$, since completion errors via SSC-Lifting and LRMC are sufficiently small, the recovered matrices will be noisy versions of the original matrices. As a result, Lasso-type optimizations of SSC and SSC-Lifting will succeed in recovering subspace-sparse representations, leading to zero clustering errors. In the high-rank case, SSC-EC has a higher clustering error than LRMC and SSC-Lifting, which is due to the fact that it relies on a heuristic of shifting eigenvalues of the kernel matrix to non-negative values.

## 5.2    Real Experiments on Motion Segmentation

We consider the problem of motion segmentation [37, 38] with missing entries on the Hopkins 155 dataset, with 155 sequences of 2 and 3 motions. Since the dataset consists of complete feature trajectories (incomplete trajectories were removed manually to form the dataset), we select $\rho$ fraction of feature points across all frames uniformly at random and remove their $x - y$ coordinate values.

Left plot in Figure 4 shows clustering error bars of different algorithms on the dataset as a function of $\rho$. Notice that in all cases, MFA and SSC-CEC have large errors, due to, respectively, dependence on initialization and the heuristic convex reformulation. On the other hand, LRMC+SSC and SSC-Lifting perform well, achieving less than 5% error for all values of $\rho$. This comes from the fact that sequences have at most $L = 3$ motions and dimension of each motion subspace is at most $d = 4$, hence, $Ld \leq 12 \ll 2F$, where $F$ is the number of video frames. Since the data matrix is low-rank and LRMC succeeds, SSC and our method achieve roughly the same errors for different values of $\rho$.

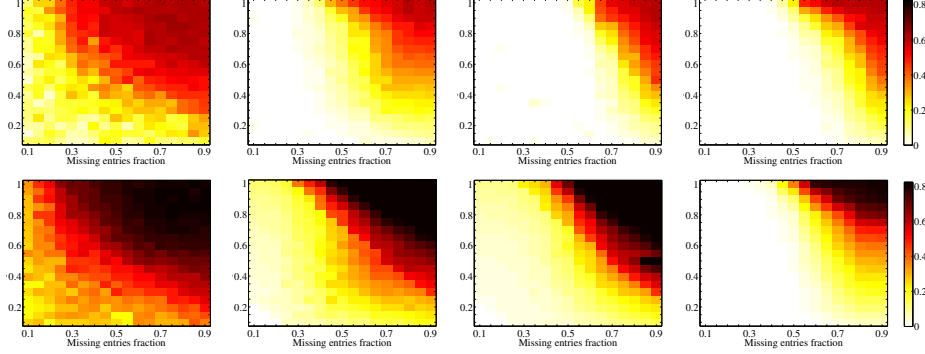

Figure 3: Clustering errors for low-rank matrices (top row) with $L = 3, d = 5, n = 100$ and high-rank matrices (bottom row) with $L = 12, d = 10, n = 100$ as a function of $(\rho, \delta)$, where $\delta$ is the fraction of data with missing entires (vertical axis) and $\rho$ is the fraction of missing entries in each affected point (horizontal axis). Left to Right: MFA, SSC-CEC, LRMC+SSC and SSC-Lifting.

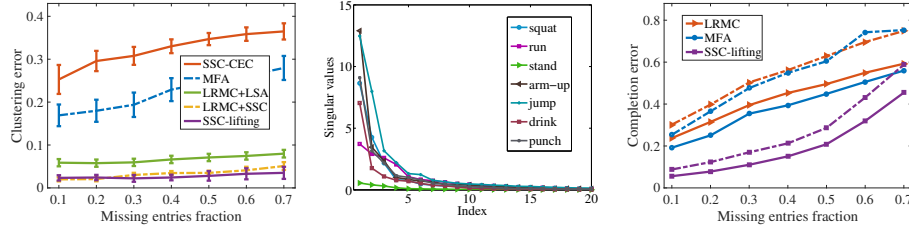

Figure 4: Left: Clustering error bars of MFA, LRMC+LSA, LRMC+SSC, SSC-CEC and SSC-Lifting as a function of the fraction of missing entries, $\rho$. Middle: Singular values of CMU Mocap data reveal that each activity lie in a low-dimensional subspace. Right: Average completion errors of MFA, LRMC and SSC-Lifting on the CMU Mocap Dataset as a function of $\rho$. Solid lines correspond to $\delta = 0.5$, i.e., $50\%$ of data have missing entries, while dashed lines correspond to $\delta = 1$, i.e., all data have missing entries.

## 5.3 Real Experiments on Motion Capture Data

We consider completion of time-series trajectories from motion capture sensors, where a trajectory consists of different human activities, such as running, jumping, squatting, etc. We use the CMU Mocap dataset, where each data point corresponds to measurements from $n$ sensors at a particular time instant. Since transition from one activity to another happens gradually, we do not consider clustering. However, as the middle plot in Figure 4 shows, excluding the transition time periods, data from each activity lie in a low-rank subspace. Since typically there are $L \approx 7$ activities, each having a dimension of $d \approx 8$, and there are $n = 42$ sensors, the data matrix is full-rank, as $Ld \approx 56 > n = 42$.

To evaluate performance of different algorithms, we select $\delta \in \{0.5, 1.0\}$ fraction of data points and remove entries of $\rho \in \{0.1, 0.2, 0.3, 0.4, 0.5, 0.6, 0.7\}$ fraction of each selected point, both uniformly at random. Right plot in Figure 4 shows completion errors of different algorithms as a function of $\rho$ for $\delta \in \{0.5, 1.0\}$. Notice that, unlike the previous experiment, since the data matrix is high-rank, LRMC has a large completion error, similar to synthetic experiments. On the other hand, SSC-Lifting error is less than 0.1 for $\rho = 0.1$ and less than 0.55 for $\rho = 0.7$. In all cases, for $\delta = 1$, the performance degrades with respect to $\delta = 0.5$. Lastly, it is important to notice that MFA performs slightly better than LRMC, demonstrating the importance of the union of low-dimensional subspaces model for the problem. However, getting trapped in local optima does not allow MFA to take full advantage of such a model, as opposed to SSC-Lifting.

## 6 Conclusions

We proposed efficient algorithms, based on lifting, for simultaneous clustering and completion of incomplete multi-subspace data. By extensive experiments on synthetic and real data, we showed that for low-rank data matrices, our algorithm performs on par with or better than low-rank matrix completion methods, while for high-rank data matrices, it significantly outperforms existing algorithms. Theoretical guarantees of the proposed method and scaling the algorithm to large data is the subject of our ongoing research.

## Footnotes

[2]$\ell_1$ is the convex surrogate of the cardinality function, $\sum_{i=1}^N \mathrm{I}(|c_{ij}|)$, where $\mathrm{I}(\cdot)$ is the indicator function.

[3]The nuclear norm of $\boldsymbol{A}$, denoted by $\|\boldsymbol{A}\|_*$, is the sum of its singular values, i.e., $\|\boldsymbol{A}\|_* = \sum_i \sigma_i(\boldsymbol{A})$.

[4]The denominator is always nonzero since $c_{ii} = -1$ for all $i$.

[5]Stable rank of $\boldsymbol{B}$ is defined as $\sum_i \sigma_i^2 / \max_i \sigma_i^2$, where $\sigma_i$'s are singular values of $\boldsymbol{B}$.

# References

[1] R. Basri and D. Jacobs, "Lambertian reflection and linear subspaces," *IEEE Transactions on Pattern Analysis and Machine Intelligence*, vol. 25, 2003.

[2] T. Hastie and P. Simard, "Metrics and models for handwritten character recognition," *Statistical Science*, 1998.

[3] C. Tomasi and T. Kanade, "Shape and motion from image streams under orthography," *International Journal of Computer Vision*, vol. 9, 1992.

[4] E. Elhamifar and R. Vidal, "Sparse subspace clustering: Algorithm, theory, and applications," *IEEE Transactions on Pattern Analysis and Machine Intelligence*, 2013.

[5] G. Chen and G. Lerman, "Spectral curvature clustering (SCC)," *International Journal of Computer Vision*, vol. 81, 2009.

[6] A. Zhang, N. Fawaz, S. Ioannidis, and A. Montanari, "Guess who rated this movie: Identifying users through subspace clustering," *Uncertainty in Artificial Intelligence (UAI)*, 2012.

[7] R. Vidal, R. Tron, and R. Hartley, "Multiframe motion segmentation with missing data using PowerFactorization and GPCA," *International Journal of Computer Vision*, vol. 79, 2008.

[8] J. Mairal, F. Bach, J. Ponce, and G. Sapiro, "Online dictionary learning for sparse coding," in *International Conference on Machine Learning*, 2009.

[9] D. Park, J. Neeman, J. Zhang, S. Sanghavi, and I. S. Dhillon, "Preference completion: Large-scale collaborative ranking from pairwise comparisons," *International Conference on Machine Learning (ICML)*, 2015.

[10] M. Tipping and C. Bishop, "Probabilistic principal component analysis," *Journal of the Royal Statistical Society*, vol. 61, 1999.

[11] M. Knott and D. Bartholomew, *Latent variable models and factor analysis*. London: Edward Arnold, 1999.

[12] E. J. Candès and B. Recht, "Exact matrix completion via convex optimization," *Foundations of Computational Mathematics*, vol. 9, 2008.

[13] E. J. Candès and Y. Plan, "Matrix completion with noise," *Proceedings of the IEEE*, 2009.

[14] R. Keshavan, A. Montanari, and S. Oh, "Matrix completion from noisy entries," *IEEE Transactions on Information Theory*, 2010.

[15] Y. Chen, H. Xu, C. Caramanis, and S. Sanghavi, "Robust matrix completion with corrupted columns," in *International Conference on Machine Learning (ICML)*, 2011.

[16] S. Bhojanapalli and P. Jain, "Universal matrix completion," *International Conference on Machine Learning (ICML)*, 2013.

[17] K. Y. Chiang, C. J. Hsieh, and I. S. Dhillon, "Matrix completion with noisy side information," *Neural Information Processing Systems (NIPS)*, 2015.

[18] M. Tipping and C. Bishop, "Mixtures of probabilistic principal component analyzers," *Neural Computation*, vol. 11, 1999.

[19] A. Gruber and Y. Weiss, "Multibody factorization with uncertainty and missing data using the em algorithm," *IEEE Conference on Computer Vision and Pattern Recognition (CVPR)*, 2004.

[20] Z. Ghahramani and G. E. Hinton, "The em algorithm for mixtures of factor analyzers," *Technical Report CRG-TR-96-1, Dept. Computer Science, Univ. of Toronto*, 1996.

[21] L. Balzano, A. Szlam, B. Recht, and R. Nowak, "K-subspaces with missing data," *IEEE Statistical Signal Processing Workshop*, 2012.

[22] B. Eriksson, L. Balzano, and R. Nowak, "High rank matrix completion," *International Conference on Artificial Intelligence and Statistics*, 2012.

[23] E. J. Candes, L. Mackey, and M. Soltanolkotabi, "From robust subspace clustering to full-rank matrix completion," *Unpublished abstract*, 2014.

[24] C. Yang, D. Robinson, and R. Vidal, "Sparse subspace clustering with missing entries," *International Conference on Machine Learning (ICML)*, 2015.

[25] R. Ganti, L. Balzano, and R. Willett, "Matrix completion under monotonic single index models," *Neural Information Processing Systems (NIPS)*, 2015.

[26] E. Elhamifar and R. Vidal, "Sparse subspace clustering," in *IEEE Conference on Computer Vision and Pattern Recognition*, 2009.

[27] A. Ng, Y. Weiss, and M. Jordan, "On spectral clustering: analysis and an algorithm," in *Neural Information Processing Systems*, 2001.

[28] M. Soltanolkotabi, E. Elhamifar, and E. J. Candes, "Robust subspace clustering," *Annals of Statistics*, 2014.

[29] B. Zhao, G. Rocha, and B. Yu, "The composite absolute penalties family for grouped and hierarchical selection," *The Annals of Statistics*, vol. 37, 2009.

[30] R. Jenatton, J. Y. Audibert, and F. Bach, "Structured variable selection with sparsity-inducing norms," *Journal of Machine Learning Research*, vol. 12, 2011.

[31] J. Tropp, "Column subset selection, matrix factorization, and eigenvalue optimization," in *ACM-SIAM Symp. Discrete Algorithms (SODA)*, 2009.

[32] E. Elhamifar, G. Sapiro, and S. S. Sastry, "Dissimilarity-based sparse subset selection," *IEEE Transactions on Pattern Analysis and Machine Intelligence*, 2016.

[33] E. Elhamifar, G. Sapiro, and R. Vidal, "See all by looking at a few: Sparse modeling for finding representative objects," in *IEEE Conference on Computer Vision and Pattern Recognition*, 2012.

[34] S. Boyd, N. Parikh, E. Chu, B. Peleato, and J. Eckstein, "Distributed optimization and statistical learning via the alternating direction method of multipliers," *Foundations and Trends in Machine Learning*, vol. 3, 2010.

[35] D. Gabay and B. Mercier, "A dual algorithm for the solution of nonlinear variational problems via finite-element approximations," *Comp. Math. Appl.*, vol. 2, 1976.

[36] J. Yan and M. Pollefeys, "A general framework for motion segmentation: Independent, articulated, rigid, non-rigid, degenerate and non-degenerate," in *European Conf. on Computer Vision*, 2006.

[37] J. Costeira and T. Kanade, "A multibody factorization method for independently moving objects." *Int. Journal of Computer Vision*, vol. 29, 1998.

[38] K. Kanatani, "Motion segmentation by subspace separation and model selection," in *IEEE Int. Conf. on Computer Vision*, vol. 2, 2001.

